# Generalised Coupled Tensor Factorisation

**Y. Kenan Yılmaz**     **A. Taylan Cemgil**     **Umut Şimşekli**
Department of Computer Engineering
Boğaziçi University, Istanbul, Turkey
kenan@sibnet.com.tr, {taylan.cemgil, umut.simsekli}@boun.edu.tr

## Abstract

We derive algorithms for generalised tensor factorisation (GTF) by building upon the well-established theory of Generalised Linear Models. Our algorithms are general in the sense that we can compute arbitrary factorisations in a message passing framework, derived for a broad class of exponential family distributions including special cases such as Tweedie's distributions corresponding to $\beta$-divergences. By bounding the step size of the Fisher Scoring iteration of the GLM, we obtain general updates for real data and multiplicative updates for non-negative data. The GTF framework is, then extended easily to address the problems when multiple observed tensors are factorised simultaneously. We illustrate our coupled factorisation approach on synthetic data as well as on a musical audio restoration problem.

## 1   Introduction

A fruitful modelling approach for extracting meaningful information from highly structured multivariate datasets is based on matrix factorisations (MFs). In fact, many standard data processing methods of machine learning and statistics such as clustering, source separation, independent components analysis (ICA), nonnegative matrix factorisation (NMF), latent semantic indexing (LSI) can be expressed and understood as MF problems. These MF models also have well understood probabilistic interpretations as probabilistic generative models. Indeed, many standard algorithms mentioned above can be derived as maximum likelihood or maximum a-posteriori parameter estimation procedures. It is also possible to do a full Bayesian treatment for model selection [1].

Tensors appear as a natural generalisation of matrix factorisation, when observed data and/or a latent representation have several semantically meaningful dimensions. Before giving a formal definition, consider the following motivating example

$$X_1^{i,j,k} \approx \sum_r Z_1^{i,r} Z_2^{j,r} Z_3^{k,r} \qquad X_2^{j,p} \approx \sum_r Z_2^{j,r} Z_4^{p,r} \qquad X_3^{j,q} \approx \sum_r Z_2^{j,r} Z_5^{q,r} \qquad (1)$$

where $X_1$ is an observed 3-way array and $X_2, X_3$ are 2-way arrays, while $Z_\alpha$ for $\alpha = 1 \ldots 5$ are the latent 2-way arrays. Here, the 2-way arrays are just matrices but this can be easily extended to objects having arbitrary number of indices. As the term '$N$-way array' is awkward, we prefer using the more convenient term *tensor*. Here, $Z_2$ is a shared factor, coupling all models. As the first model is a CP (Parafac) while the second and the third ones are MF's, we call the combined factorization as CP/MF/MF model. Such models are of interest when one can obtain different 'views' of the same piece of information (here $Z_2$) under different experimental conditions. Singh and Gordon [2] focused on a similar problem called as *collective matrix factorisation* (CMF) or *multi-matrix factorisation*, for relational learning but only for matrix factors and observations. In addition, their generalised Bregman divergence minimisation procedure assumes matching link and loss functions. For *coupled matrix and tensor factorization* (CMTF), recently [3] proposed a gradient-based all-at-once optimization method as an alternative to *alternating least square* (ALS) optimization and

demonstrated their approach for a CP/MF coupled model. Similar models are used for protein-protein interactions (PPI) problems in gene regulation [4].

The main motivation of the current paper is to construct a general and practical framework for computation of tensor factorisations (TF), by extending the well-established theory of Generalised Linear Models (GLM). Our approach is also partially inspired by probabilistic graphical models: our computation procedures for a given factorisation have a natural message passing interpretation. This provides a structured and efficient approach that enables very easy development of application specific custom models, priors or error measures as well as algorithms for joint factorisations where an arbitrary set of tensors can be factorised simultaneously. Well known models of multiway analysis (Parafac, Tucker [5]) appear as special cases and novel models and associated inference algorithms can be automatically be developed. In [6], the authors take a similar approach to tensor factorisations as ours, but that work is limited to $KL$ and Euclidean costs, generalising MF models of [7] to the tensor case. It is possible to generalise this line of work to $\beta$-divergences [8] but none of these works address the coupled factorisation case and consider only a restricted class of cost functions.

## 2    Generalised Linear Models for Matrix/Tensor Factorisation

To set the notation and our approach, we briefly review GLMs following closely the original notation of [9, ch 5]. A GLM assumes that a data vector $x$ has conditionally independently drawn components $x_i$ according to an exponential family density

$$x_i \sim \exp\left(\frac{x_i \gamma_i - b(\gamma_i)}{\tau^2} - c(x_i, \tau)\right) \qquad \langle x_i \rangle = \hat{x}_i = \frac{\partial b(\gamma_i)}{\partial \gamma_i} \qquad var(x_i) = \tau^2 \frac{\partial^2 b(\gamma_i)}{\partial \gamma_i^2} \qquad (2)$$

Here, $\gamma_i$ are *canonical parameters* and $\tau^2$ is a known dispersion parameter. $\langle x_i \rangle$ is the expectation of $x_i$ and $b(\cdot)$ is the log partition function, enforcing normalization. The canonical parameters are not directly estimated, instead one assumes a link function $g(\cdot)$ that 'links' the mean of the distribution $\hat{x}_i$ and assumes that $g(\hat{x}_i) = l_i^\top z$ where $l_i^\top$ is the $i$th row vector of a known model matrix $L$ and $z$ is the parameter vector to be estimated, $A^\top$ denotes matrix transpose of $A$. The model is linear in the sense that a function of the mean is linear in parameters, i.e., $g(\hat{x}) = Lz$ . A *Linear Model (LM)* is a special case of GLM that assumes normality, i.e. $x_i \sim \mathcal{N}(x_i; \hat{x}_i, \sigma^2)$ as well as linearity that implies identity link function as $g(\hat{x}_i) = \hat{x}_i = l_i^\top z$ assuming $l_i$ are known. Logistic regression assumes a log link, $g(\hat{x}_i) = \log \hat{x}_i = l_i^\top z$; here $\log \hat{x}_i$ and $z$ have a linear relationship [9].

The goal in classical GLM is to estimate the parameter vector $z$. This is typically achieved via a Gauss-Newton method (Fisher Scoring). The necessary objects for this computation are the log likelihood, the derivative and the Fisher Information (the expected value of negative of the Fisher Score). These are easily derived as:

$$\mathcal{L} = \sum_i [x_i \gamma_i - b(\gamma_i)]/\tau^2 - \sum_i c(x_i, \tau) \qquad \frac{\partial \mathcal{L}}{\partial z} = \frac{1}{\tau^2} \sum_i (x_i - \hat{x}_i) w_i g_{\hat{x}}(\hat{x}_i) l_i^\top \qquad (3)$$

$$\frac{\partial \mathcal{L}}{\partial z} = \frac{1}{\tau^2} L^\top DG(x - \hat{x}) \qquad \left\langle \frac{\partial^2 \mathcal{L}}{\partial z^2} \right\rangle = \frac{1}{\tau^2} L^\top DL \qquad (4)$$

where $w$ is a vector with elements $w_i$, $D$ and $G$ are the diagonal matrices as $D = diag(w)$, $G = diag(g_{\hat{x}}(\hat{x}_i))$ and

$$w_i = \left(v(\hat{x}_i) g_{\hat{x}}^2(\hat{x}_i)\right)^{-1} \qquad g_{\hat{x}}(\hat{x}_i) = \frac{\partial g(\hat{x}_i)}{\partial \hat{x}_i} \qquad (5)$$

with $v(\hat{x}_i)$ being the *variance function* related to the observation variance by $var(x_i) = \tau^2 v(\hat{x}_i)$. Via Fisher Scoring, the general update equation in matrix form is written as

$$z \leftarrow z + \left(L^\top DL\right)^{-1} L^\top DG(x - \hat{x}) \qquad (6)$$

Although this formulation is somewhat abstract, it covers a very broad range of model classes that are used in practice. For example, an important special case appears when the variance functions are in the form of $v(\hat{x}) = \hat{x}^p$. By setting $p = \{0, 1, 2, 3\}$ these correspond to Gaussian, Poisson, Exponential/Gamma, and Inverse Gaussian distributions [10, pp.30], which are special cases of the exponential family of distributions for any $p$ named Tweedie's family [11]. Those for $p = \{0, 1, 2\}$, in turn, correspond to EU, KL and IS cost functions often used for NMF decompositions [12, 7].

## 2.1 Tensor Factorisations (TF) as GLM's

The key observation for expressing a TF model as a GLM is to identify the multilinear structure and using an alternating optimization approach. To hide the notational complexity, we will give an example with a simple matrix factorisation model; extension to tensors will require heavier notation, but are otherwise conceptually straightforward. Consider a MF model

$$g(\hat{X}) = Z_1 Z_2 \qquad \text{in scalar} \qquad g(\hat{X})^{i,j} = \sum_r Z_1^{i,r} Z_2^{j,r} \qquad (7)$$

where $Z_1, Z_2$ and $g(\hat{X})$ are matrices of compatible sizes. Indeed, by applying the **vec** operator (vectorization, stacking columns of a matrix to obtain a vector) to both sides of (7) we obtain two equivalent representation of the same system

$$\mathbf{vec}(g(\hat{X})) = (I_{|j|} \otimes Z_1)\, \mathbf{vec}(Z_2) = \frac{\partial(Z_1 Z_2)}{\partial Z_2}\, \mathbf{vec}(Z_2) = \frac{\partial g(\hat{X})}{\partial Z_2}\, \mathbf{vec}(Z_2) \equiv \nabla_2 \vec{Z}_2 \qquad (8)$$

where $I_{|j|}$ denotes the $|j| \times |j|$ identity matrix, $\otimes$ denotes the Kronecker product [13], and **vec** $Z \equiv \vec{Z}$. Clearly, this is a GLM where $\nabla_2$ plays the role of a model matrix and $\vec{Z}_2$ is the parameter vector. By alternating between $Z_1$ and $Z_2$, we can maximise the log-likelihood iteratively; indeed this alternating maximisation is standard for solving matrix factorisation problems. In the sequel, we will show that a much broader range of algorithms can be readily derived in the GLM framework.

## 2.2 Generalised Tensor Factorisation

We define a *tensor* $\Lambda$ as a multiway array with an index set $\mathcal{V} = \{i_1, i_2, \ldots, i_{|\alpha|}\}$ where each index $i_n$ for $n = 1 \ldots |\alpha|$ runs as $i_n = 1 \ldots |i_n|$. An *element of the tensor* $\Lambda$ is a scalar that we denote by $\Lambda(i_1, i_2, \ldots, i_{|\alpha|})$ or $\Lambda^{i_1, i_2, \ldots, i_{|\alpha|}}$ or as a shorthand notation by $\Lambda(v)$ with $v$ being a particular configuration. $|v|$ denotes number of all distinct configurations for $\mathcal{V}$, and e.g. if $\mathcal{V} = \{i_1, i_2\}$ then $|v| = |i_1||i_2|$. We call the form $\Lambda(v)$ as *element-wise*; the notation $[\ ]$ yields a tensor by enumerating all the indices, i.e., $\Lambda = [\Lambda^{i_1, i_2, \ldots, i_{|\alpha|}}]$ or $\Lambda = [\Lambda(v)]$. For any two tensors $X$ and $Y$ of compatible order, $X \circ Y$ is an element-wise multiplication and if not explicitly stressed $X/Y$ is an element-wise division. $\mathbf{1}$ is an object of all ones whose order depends on the context where it is used.

A generalised tensor factorisation problem is specified by an observed tensor $X$ (with possibly missing entries, to be treated later) and a *collection of latent tensors* to be estimated, $Z_{1:|\alpha|} = \{Z_\alpha\}$ for $\alpha = 1 \ldots |\alpha|$, and by an exponential family of form (2). The index set of $X$ is denoted by $\mathcal{V}_0$ and the index set of each $Z_\alpha$ by $\mathcal{V}_\alpha$. The set of all model indices is $\mathcal{V} = \bigcup_{\alpha=1}^{|\alpha|} \mathcal{V}_\alpha$. We use $v_\alpha$ (or $v_0$) to denote a particular configuration of the indices for $Z_\alpha$ (or $X$) while $\bar{v}_\alpha$ denoting a configuration of the compliment $\bar{\mathcal{V}}_\alpha = \mathcal{V}/\mathcal{V}_\alpha$. The goal is to find the latent $Z_\alpha$ that maximize the likelihood $p(X|Z_{1:\alpha})$ where $\langle X \rangle = \hat{X}$ is given via

$$g(\hat{X}(v_0)) = \sum_{\bar{v}_0} \prod_\alpha Z_\alpha(v_\alpha) \qquad (9)$$

To clarify our notation with an example, we express the CP (Parafac) model, defined as $\hat{X}(i, j, k) = \sum_r Z_1(i, r) Z_2(j, r) Z_3(k, r)$. In our notation, we take identity link $g(\hat{X}) = \hat{X}$ and the index sets with $\mathcal{V} = \{i, j, k, r\}$, $\mathcal{V}_0 = \{i, j, k\}$, $\bar{\mathcal{V}}_0 = \{r\}$, $\mathcal{V}_1 = \{i, r\}$, $\mathcal{V}_2 = \{j, r\}$ and $\mathcal{V}_3 = \{k, r\}$. Our notation deliberately follows that of graphical models; the reader might find it useful to associate indices with discrete random variables and factors with probability tables [14]. Obviously, while a TF model does not represent a discrete probability measure, the algebraic structure is nevertheless analogous.

To extend the discussion in Section 2.1 to the tensor case, we need the equivalent of the model matrix, when updating $Z_\alpha$. This is obtained by summing over the product of all remaining factors

$$g(\hat{X}(v_0)) = \sum_{\bar{v}_0 \cap v_\alpha} Z_\alpha(v_\alpha) \sum_{\bar{v}_0 \cap \bar{v}_\alpha} \prod_{\alpha' \neq \alpha} Z_{\alpha'}(v_{\alpha'}) = \sum_{\bar{v}_0 \cap v_\alpha} Z_\alpha(v_\alpha) L_\alpha(o_\alpha)$$

$$L_\alpha(o_\alpha) = \sum_{\bar{v}_0 \cap \bar{v}_\alpha} \prod_{\alpha' \neq \alpha} Z_{\alpha'}(v_{\alpha'}) \qquad \text{with } o_\alpha \equiv (v_0 \cup v_\alpha) \cap (\bar{v}_0 \cup \bar{v}_\alpha)$$

One related quantity to $L_\alpha$ is the derivative of the tensor $g(\hat{X})$ wrt the latent tensor $Z_\alpha$ denoted as $\nabla_\alpha$ and is defined as (following the convention [13, pp 196])

$$\nabla_\alpha = \frac{\partial g(\hat{X})}{\partial Z_\alpha} = I_{|v_0 \cap v_\alpha|} \otimes L_\alpha \qquad \text{with } L_\alpha \in \mathbb{R}^{|v_0 \cap \bar{v}_\alpha| \times |\bar{v}_0 \cap v_\alpha|} \tag{10}$$

The importance of $L_\alpha$ is that, all the update rules can be formulated by a product and subsequent contraction of $L_\alpha$ with another tensor $Q$ having exactly the same index set of the observed tensor $X$. As a notational abstraction, it is useful to formulate the following function,

**Definition 1.** *The tensor valued function $\Delta_\alpha(Q) : \mathbb{R}^{|v_0|} \to \mathbb{R}^{|v_\alpha|}$ is defined as*

$$\Delta_\alpha^\varepsilon(Q) = \left[ \sum_{v_0 \cap \bar{v}_\alpha} Q(v_0) \, L_\alpha(o_\alpha)^\varepsilon \right] \tag{11}$$

with $\Delta_\alpha(Q)$ being an object of the same order as $Z_\alpha$ and $o_\alpha \equiv (v_0 \cup v_\alpha) \cap (\bar{v}_0 \cup \bar{v}_\alpha)$. Here, on the right side, the nonnegative integer $\varepsilon$ denotes the element-wise power, not to be confused with an index. On the left, it should be interpreted as a parameter of the $\Delta$ function. Arguably, $\Delta$ function abstracts away all the tedious reshape and unfolding operations [5]. This abstraction has also an important practical facet: the computation of $\Delta$ is algebraically (almost) equivalent to computation of marginal quantities on a factor graph, for which efficient message passing algorithms exist [14].

**Example 1.** *TUCKER3 is defined as* $\hat{X}^{i,j,k} = \sum_{p,q,r} A^{i,p} B^{j,q} C^{k,r} G^{p,q,r}$ *with* $\mathcal{V} = \{i,j,k,p,q,r\}$, $\mathcal{V}_0 = \{i,j,k\}$, $\mathcal{V}_A = \{i,p\}$, $\mathcal{V}_B = \{j,q\}$, $\mathcal{V}_C = \{k,r\}$, $\mathcal{V}_G = \{p,q,r\}$. *Then for the first factor A, the objects $L_A$ and $\Delta_A^\varepsilon()$ are computed as follows*

$$L_A = \left[ \sum_{q,r} B^{j,q} C^{k,r} G^{p,q,r} \right] = \left[ ((C \otimes B) G^\top)_{k,j}^p \right] = \left[ (L_A)_{k,j}^p \right] \tag{12}$$

$$\Delta_A^\varepsilon(Q) = \left[ \sum_{j,k} Q_i^{k,j} \left( L_A^\varepsilon \right)_{k,j}^p \right] = \left[ (Q L_A^\varepsilon)_i^p \right] \tag{13}$$

*The index sets marginalised out for $L_A$ and $\Delta_A$ are $\bar{\mathcal{V}}_0 \cap \bar{\mathcal{V}}_A = \{p,q,r\} \cap \{j,q,k,r\} = \{q,r\}$ and $\mathcal{V}_0 \cap \bar{\mathcal{V}}_A = \{i,j,k\} \cap \{j,q,k,r\} = \{j,k\}$. Also we verify the order of the gradient $\nabla_A$ (10) as $I_i^i \otimes L_{A_{k,j}}^p = \nabla_{i,k,j}^{i,p}$ that conforms the matrix derivation convention [13, pp.196].*

## 2.3 Iterative Solution for GTF

As we have now established a one to one relationship between GLM and GTF objects such as the observation $x \equiv \text{vec X}$, the mean (and the model estimate) $\hat{x} \equiv \text{vec } \hat{X}$, the model matrix $L \equiv L_\alpha$ and the parameter vector $z \equiv \text{vec } Z_\alpha$, we can write directly from (6) as

$$\vec{Z}_\alpha \leftarrow \vec{Z}_\alpha + \left( \nabla_\alpha^\top D \nabla_\alpha \right)^{-1} \nabla_\alpha^\top D G (\vec{X} - \vec{\hat{X}}) \qquad \text{with } \nabla_\alpha = \frac{\partial g(\hat{X})}{\partial Z_\alpha} \tag{14}$$

There are at least two ways that this update can further simplified. We may assume an identity link function, or alternatively we may choose a matching link and lost functions such that they cancel each other smoothly [2]. In the sequel we consider identity link $g(\hat{X}) = \hat{X}$ that results to $g_{\hat{X}}(\hat{X}) = \mathbf{1}$. This implies $G$ to be identity, i.e. $G = I$. We define a tensor $W$, that plays the same role as $w$ in (5), which becomes simply the precision (inverse variance function), i.e. $W = 1/v(\hat{X})$ where for the Gaussian, Poisson, Exponential and Inverse Gaussian distributions we have simply $W = \hat{X}^{-p}$ with $p = \{0,1,2,3\}$ [10, pp 30]. Then, the update (14) is reduced to

$$\vec{Z}_\alpha \leftarrow \vec{Z}_\alpha + \left( \nabla_\alpha^\top D \nabla_\alpha \right)^{-1} \nabla_\alpha^\top D (\vec{X} - \vec{\hat{X}}) \tag{15}$$

After this simplification we obtain two update rules for GTF for non-negative and real data.

The update (15) can be used to derive multiplicative update rules (MUR) popularised by [15] for the nonnegative matrix factorisation (NMF). MUR equations ensure the non-negative parameter updates as long as starting some non-negative initial values.

**Theorem 1.** *The update equation* (15) *for nonnegative GTF is reduced to multiplicative form as*

$$Z_\alpha \leftarrow Z_\alpha \circ \frac{\Delta_\alpha(W \circ X)}{\Delta_\alpha(W \circ \hat{X})} \qquad\qquad s.t.\ Z_\alpha(v_\alpha) > 0 \qquad (16)$$

**(Proof sketch)** Due to space limitation we leave the full details of the proof, but idea is that inverse of $H = \nabla^\top D \nabla$ is identified as step size and by use of the results of the Perron-Frobenious theorem [16, pp 125] we further bound it as

$$\eta = \frac{\vec{Z}_\alpha}{\nabla^\top D \vec{\hat{X}}} < \frac{2\vec{Z}_\alpha}{\nabla^\top D \vec{\hat{X}}} \leq \frac{2}{\lambda_{max}(\nabla^\top D \nabla)} \qquad \text{since } \lambda_{max}(H) \leq \max_{v_\alpha} \frac{(H\vec{Z}_\alpha)(v_\alpha)}{Z_\alpha(v_\alpha)} \qquad (17)$$

For the special case of the Tweedie family where the precision is a function of the mean as $W = \hat{X}^{-p}$ for $p = \{0, 1, 2, 3\}$ the update (15) is reduced to

$$Z_\alpha \leftarrow Z_\alpha \circ \frac{\Delta_\alpha(\hat{X}^{-p} \circ X)}{\Delta_\alpha(\hat{X}^{1-p})} \qquad\qquad (18)$$

For example, to update $Z_2$ for the NMF model $\hat{X} = Z_1 Z_2$, $\Delta_2$ is $\Delta_2(Q) = Z_1^\top Q$. Then for the Gaussian ($p = 0$) this reduces to NMF-EU as $Z_2 \leftarrow Z_2 \circ (Z_1^\top X)/(Z_1^\top \hat{X})$. For the Poisson ($p = 1$) it reduces to NMF-KL as $Z_2 \leftarrow Z_2 \circ \left(Z_1^\top(X/\hat{X})\right)/\left(Z_1^\top \mathbf{1}\right)$ [15].

By dropping the non-negativity requirement we obtain the following update equation:

**Theorem 2.** *The update equation for GTF with real data can be expressed as*

$$Z_\alpha \leftarrow Z_\alpha + \frac{2}{\lambda_{\alpha/0}} \frac{\Delta_\alpha(W \circ (X - \hat{X}))}{\Delta_\alpha^2(W)} \qquad\qquad \text{with } \lambda_{\alpha/0} = |v_\alpha \cap \bar{v}_0| \qquad (19)$$

**(Proof sketch)** Again skipping the full details, as part of the proof we set $Z_\alpha = \mathbf{1}$ in (17) specifically, and replacing matrix multiplication of $\nabla^\top D \nabla \mathbf{1}$ by $\nabla^{\top 2} D \mathbf{1} \lambda_{\alpha/0}$ completes the proof. Here the multiplier $\lambda_{\alpha/0}$ is the cardinality arising from the fact that only $\lambda_{\alpha/0}$ elements are non-zero in a row of $\nabla^\top D \nabla$. Note the example for $\lambda_{\alpha/0}$ that if $\mathcal{V}_\alpha \cap \bar{\mathcal{V}}_0 = \{p, q\}$ then $\lambda_{\alpha/0} = |p||q|$ which is number of all distinct configurations for the index set $\{p, q\}$.

**Missing data** can be handled easily by dropping the missing data terms from the likelihood [17]. The net effect of this is the addition of an indicator variable $m_i$ to the gradient $\partial\mathcal{L}/\partial z = \tau^{-2} \sum_i(x_i - \hat{x}_i)m_i w_i g_{\hat{x}}(\hat{x}_i)l_i^\top$ with $m_i = 1$ if $x_i$ is observed otherwise $m_i = 0$. Hence we simply define a mask tensor $M$ having the same order as the observation $X$, where the element $M(v_0)$ is 1 if $X(v_0)$ is observed and zero otherwise. In the update equations, we merely replace $W$ with $W \circ M$.

## 3   Coupled Tensor Factorization

Here we address the problem when multiple observed tensors $X_\nu$ for $\nu = 1 \ldots |\nu|$ are factorised simultaneously. Each observed tensor $X_\nu$ now has a corresponding index set $\mathcal{V}_{0,\nu}$ and a particular configuration will be denoted by $v_{0,\nu} \equiv u_\nu$. Next, we define a $|\nu| \times |\alpha|$ *coupling matrix* $R$ where

$$R^{\nu,\alpha} = \begin{cases} 1 & X_\nu \text{ and } Z_\alpha \text{ connected} \\ 0 & \text{otherwise} \end{cases} \qquad\qquad \hat{X}_\nu(u_\nu) = \sum_{\bar{u}_\nu} \prod_\alpha Z_\alpha(v_\alpha)^{R^{\nu,\alpha}} \qquad (20)$$

For the coupled factorisation, we get the following expression as the derivative of the log likelihood

$$\frac{\partial\mathcal{L}}{\partial Z_\alpha(v_\alpha)} = \sum_\nu R^{\nu,\alpha} \sum_{u_\nu \cap \bar{v}_\alpha} \left(X_\nu(u_\nu) - \hat{X}_\nu(u_\nu)\right) W_\nu(u_\nu) \frac{\partial \hat{X}_\nu(u_\nu)}{\partial Z_\alpha(v_\alpha)} \qquad (21)$$

where $W_\nu \equiv W(\hat{X}_\nu(u_\nu))$ are the precisions. Then proceeding as in section 2.3 (i.e. getting the Hessian and finding Fisher Information) we arrive at the update rule in vector form as

$$\vec{Z}_\alpha \leftarrow \vec{Z}_\alpha + \left(\sum_\nu R^{\nu,\alpha} \nabla_{\alpha,\nu}^\top D_\nu \nabla_{\alpha,\nu}\right)^{-1} \left(\sum_\nu R^{\nu,\alpha} \nabla_{\alpha,\nu}^\top D_\nu (\vec{X}_\nu - \vec{\hat{X}}_\nu)\right) \qquad (22)$$

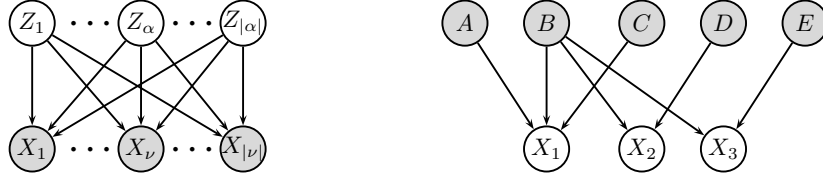

Figure 1: (Left) Coupled factorisation structure where the arrow indicates the existence of the influence of latent tensor $Z_\alpha$ onto the observed tensor $X_\nu$. (Right) The CP/MF/MF coupled factorisation problem in 1.

where $\nabla_{\alpha,\nu} = \partial g(\hat{X}_\nu)/\partial Z_\alpha$. The update equations for the coupled case are quite intuitive; we calculate the $\Delta_{\alpha,\nu}$ functions defined as

$$\Delta_{\alpha,\nu}^{\varepsilon}(Q) = \Big[ \sum_{u_\nu \cap \bar{v}_\alpha} Q(u_\nu)\Big( \prod_{\alpha' \neq \alpha} Z_{\alpha'}(v_{\alpha'})^{R^{\nu,\alpha}}\Big)^{\varepsilon}\Big] \tag{23}$$

for each submodel and add the results:

**Lemma 1.** *Update for non-negative CTF*

$$Z_\alpha \leftarrow Z_\alpha \circ \frac{\sum_\nu R^{\nu,\alpha}\Delta_{\alpha,\nu}(W_\nu \circ X_\nu)}{\sum_\nu R^{\nu,\alpha}\Delta_{\alpha,\nu}\Big(W_\nu \circ \hat{X}_\nu\Big)} \tag{24}$$

In the special case of a Tweedie family, i.e. for the distributions whose precision as $W_\nu = \hat{X}_\nu^{-p}$, the update is $Z_\alpha \leftarrow Z_\alpha \circ \Big(\sum_\nu R^{\nu,\alpha}\Delta_{\alpha,\nu}\Big(\hat{X}_\nu^{-p} \circ X_\nu\Big)\Big) / \Big(\sum_\nu R^{\nu,\alpha}\Delta_{\alpha,\nu}\Big(\hat{X}_\nu^{1-p}\Big)\Big)$.

**Lemma 2.** *General update for CTF*

$$Z_\alpha \leftarrow Z_\alpha + \frac{2}{\lambda_{\alpha/0}}\frac{\sum_\nu R^{\nu,\alpha}\Delta_{\alpha,\nu}\Big(W_\nu \circ \big(X_\nu - \hat{X}_\nu\big)\Big)}{\sum_\nu R^{\nu,\alpha}\Delta_{\alpha,\nu}^2(W_\nu)} \tag{25}$$

For the special case of the Tweedie family we plug $W_\nu = \hat{X}_\nu^{-p}$ and get the related formula.

## 4   Experiments

Here we want to solve the CTF problem introduced (1), which is a coupled CP/MF/MF problem

$$\hat{X}_1^{i,j,k} = \sum_r A^{i,r}B^{j,r}C^{k,r} \qquad \hat{X}_2^{j,p} = \sum_r B^{j,r}D^{p,r} \qquad \hat{X}_3^{j,q} = \sum_r B^{j,r}E^{q,r} \tag{26}$$

where we employ the symbols $A : E$ for the latent tensors instead of $Z_\alpha$. This factorisation problem has the following $R$ matrix with $|\alpha| = 5$, $|\nu| = 3$

$$R = \begin{bmatrix} 1 & 1 & 1 & 0 & 0 \\ 0 & 1 & 0 & 1 & 0 \\ 0 & 1 & 0 & 0 & 1 \end{bmatrix} \qquad \text{with} \quad \begin{aligned} \hat{X}_1 &= \sum A^1 B^1 C^1 D^0 E^0 \\ \hat{X}_2 &= \sum A^0 B^1 C^0 D^1 E^0 \\ \hat{X}_3 &= \sum A^0 B^1 C^0 D^0 E^1 \end{aligned} \tag{27}$$

We want to use the general update equation (25). This requires derivation of $\Delta_{\alpha,\nu}^{\varepsilon}()$ for $\nu = 1$ (CP) and $\nu = 2$ (MF) but not for $\nu = 3$ since that $\Delta_{\alpha,3}()$ has the same shape as $\Delta_{\alpha,2}()$. Here we show the computation for $B$, i.e. for $Z_2$, which is the common factor

$$\Delta_{B,1}^{\varepsilon}(Q) = \Big[\sum_{ik} Q^{i,j,k}\Big(A^{i,r}C^{k,r}\Big)^{\varepsilon}\Big] = Q_{(1)}(C^\varepsilon \odot A^\varepsilon) \tag{28}$$

$$\Delta_{B,2}^{\varepsilon}(Q) = \Big[\sum_p Q^{j,p}\big(D^{p,r}\big)^{\varepsilon}\Big] = QD^\varepsilon \tag{29}$$

with $Q_{(n)}$ being *mode*-n unfolding operation that turns a tensor into matrix form [5]. In addition, for $\nu = 1$ the required scalar value $\lambda_{B/0}$ is $|r|$ here since $\mathcal{V}_B \cap \bar{\mathcal{V}}_0 = \{j, r\} \cap \{r\} = \{r\}$ noting that value $\lambda_{B/0}$ is the same for $\nu = 2, 3$. The simulated data size for observables is $|i| = |j| = |k| = |p| = |q| = 30$ while the latent dimension is $|r| = 5$. The number of iterations is 1000 with the Euclidean cost while the experiment produced similar results for KL cost as shown in Figure 2.

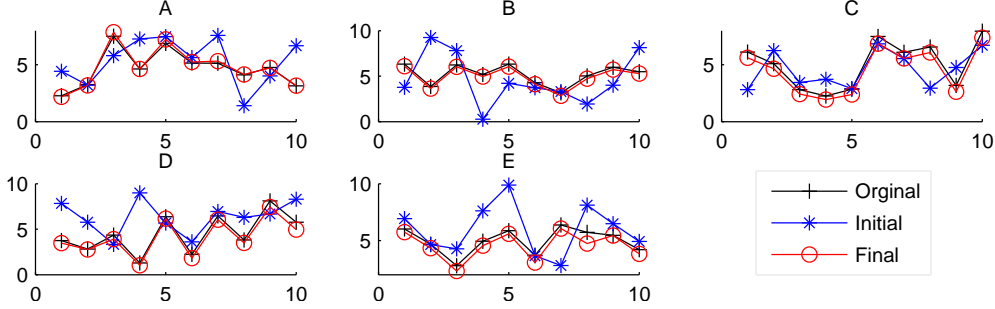

Figure 2: The figure compares the original, the initial (start up) and the final (estimate) factors for $Z_\alpha = A, B, C, D, E$. Only the first column, i.e. $Z_\alpha(1:10, 1)$ is plotted. Note that CP factorisation is unique up to permutation and scaling [5] while MF factorisation is not unique, but when coupled with CP it recovers the original data as shown in the figure. For visualisation, to find the correct permutation, for each of $Z_\alpha$ the matching permutation between the original and estimate are found by solving an *orthogonal Procrustes problem* [18, pp 601].

## 4.1   Audio Experiments

In this section, we illustrate a real data application of our approach, where we reconstruct missing parts of an audio spectrogram $X(f, t)$, that represents the STFT coefficient magnitude at frequency bin $f$ and time frame $t$ of a piano piece, see top left panel of Fig.3. This is a difficult matrix completion problem: as entire time frames (columns of $X$) are missing, low rank reconstruction techniques are likely to be ineffective. Yet such missing data patterns arise often in practice, e.g., when packets are dropped during digital communication. We will develop here a novel approach, expressed as a coupled TF model. In particular, the reconstruction will be aided by an approximate musical score, not necessarily belonging to the played piece, and spectra of isolated piano sounds.

Pioneering work of [19] has demonstrated that, when a audio spectrogram of music is decomposed using NMF as $X_1(f, t) \approx \hat{X}(f, t) = \sum_i D(f, i) E(i, t)$, the computed factors $D$ and $E$ tend to be semantically meaningful and correlate well with the intuitive notion of spectral templates (harmonic profiles of musical notes) and a musical score (reminiscent of a piano roll representation such as a MIDI file). However, as time frames are modeled conditionally independently, it is impossible to reconstruct audio with this model when entire time frames are missing.

In order to restore the missing parts in the audio, we form a model that can incorporates musical information of chords structures and how they evolve in time. In order to achieve this, we hierarchically decompose the excitation matrix $E$ as a convolution of some basis matrices and their weights: $E(i, t) = \sum_{k,\tau} B(i, \tau, k) C(k, t - \tau)$. Here the basis tensor $B$ encapsulates both vertical and temporal information of the notes that are likely to be used in a musical piece; the musical piece to be reconstructed will share $B$, possibly played at different times or tempi as modelled by $G$. After replacing $E$ with the decomposed version, we get the following model (eq 30):

$$\hat{X}_1(f, t) = \sum_{i, \tau, k, d} D(f, i) B(i, \tau, k) C(k, d) Z(d, t, \tau) \qquad \text{Test file} \qquad (30)$$

$$\hat{X}_2(i, n) = \sum_{\tau, k, m} B(i, \tau, k) G(k, m) Y(m, n, \tau) \qquad \text{MIDI file} \qquad (31)$$

$$\hat{X}_3(f, p) = \sum_i D(f, i) F(i, p) T(i, p) \qquad \text{Merged training files} \qquad (32)$$

Here we have introduced new dummy indices $d$ and $m$, and new (fixed) factors $Z(d,t,\tau) = \delta(d - t + \tau)$ and $Y(m,n,\tau) = \delta(m - n + \tau)$ to express this model in our framework. In eq 32, while forming $X_3$ we concatenate isolated recordings corresponding to different notes. Besides, $T$ is a $0 - 1$ matrix, where $T(i,p) = 1(0)$ if the note $i$ is played (not played) during the time frame $p$ and $F$ models the time varying amplitudes of the training data. $R$ matrix for this model is defined as

$$
R = \begin{bmatrix} 1 & 1 & 1 & 1 & 0 & 0 & 0 & 0 \\ 0 & 1 & 0 & 0 & 1 & 1 & 0 & 0 \\ 1 & 0 & 0 & 0 & 0 & 0 & 1 & 1 \end{bmatrix} \quad \text{with} \quad \begin{aligned} \hat{X}_1 &= \sum D^1 B^1 C^1 Z^1 G^0 Y^0 F^0 T^0 \\ \hat{X}_2 &= \sum D^0 B^1 C^0 Z^0 G^1 Y^1 F^0 T^0 \\ \hat{X}_3 &= \sum D^1 B^0 C^0 Z^0 G^0 Y^0 F^1 T^1 \end{aligned} \quad (33)
$$

Figure 3 illustrates the performance the model, using $KL$ cost ($W = \hat{X}^{-1}$) on a 30 second piano recording where the 70% of the data is missing; we get about 5dB SNR improvement, gracefully degrading from 10% to 80% missing data: the results are encouraging as quite long portions of audio are missing, see bottom right panel of Fig.3.

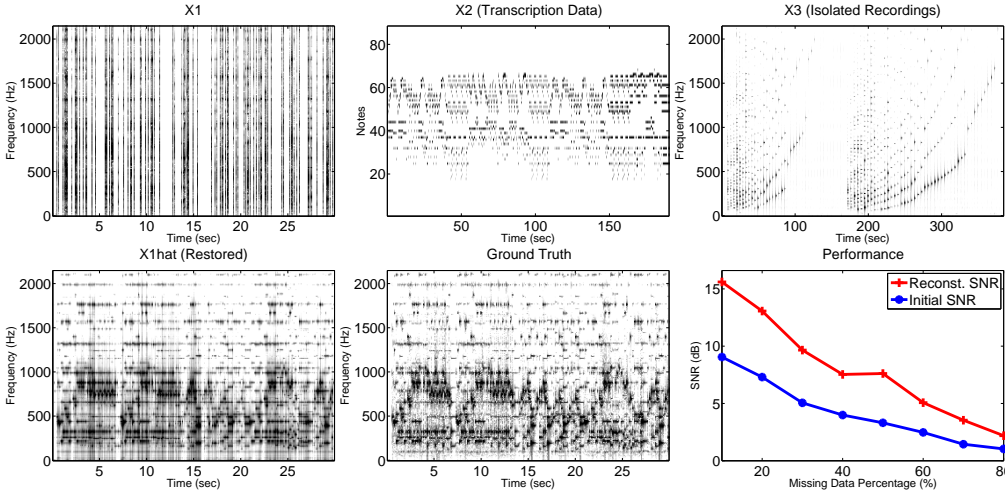

Figure 3: Top row, left to right: Observed matrices $X_1$: spectrum of the piano performance, darker colors imply higher magnitude (missing data (70%) are shown white), $X_2$, a piano roll obtained from a musical score of the piece, $X_3$, spectra of 88 isolated notes from a piano. Bottom Row: Reconstructed $X_1$, the ground truth, and the SNR results with increasing missing data. Here, initial SNR is computed by substituting 0 as missing values.

## 5 Discussion

This paper establishes a link between GLMs and TFs and provides a general solution for the computation of arbitrary coupled TFs, using message passing primitives. The current treatment focused on ML estimation; as immediate future work, the probabilistic interpretation is to be extended to a full Bayesian inference with appropriate priors and inference methods. A powerful aspect, which we have not been able to summarize here is assigning different cost functions, i.e. distributions, to different observation tensors in a coupled factorization model. This requires only minor modifications to the update equations. We believe that, as a whole, the GCTF framework covers a broad range of models that can be useful in many different application areas beyond audio processing, such as network analysis, bioinformatics or collaborative filtering.

**Acknowledgements:** This work is funded by the TÜBİTAK grant number 110E292, Bayesian matrix and tensor factorisations (BAYTEN) and Boğaziçi University research fund BAP5723. Umut Şimşekli is also supported by a Ph.D. scholarship from TÜBİTAK. We also would like to thank to Evrim Acar for the fruitful discussions.

# References

[1] A. T. Cemgil, Bayesian inference for nonnegative matrix factorisation models, Computational Intelligence and Neuroscience 2009 (2009) 1–17.

[2] A. P. Singh, G. J. Gordon, A unified view of matrix factorization models, in: ECML PKDD'08, Part II, no. 5212, Springer, 2008, pp. 358–373.

[3] E. Acar, T. G. Kolda, D. M. Dunlavy, All-at-once optimization for coupled matrix and tensor factorizations, CoRR abs/1105.3422. arXiv:1105.3422.

[4] Q. Xu, E. W. Xiang, Q. Yang, Protein-protein interaction prediction via collective matrix factorization, in: In Proc. of the IEEE International Conference on BIBM, 2010, pp. 62–67.

[5] T. G. Kolda, B. W. Bader, Tensor decompositions and applications, SIAM Review 51 (3) (2009) 455–500.

[6] Y. K. Yılmaz, A. T. Cemgil, Probabilistic latent tensor factorization, in: Proceedings of the 9th international conference on Latent variable analysis and signal separation, LVA/ICA'10, Springer-Verlag, 2010, pp. 346–353.

[7] C. Fevotte, A. T. Cemgil, Nonnegative matrix factorisations as probabilistic inference in composite models, in: Proc. 17th EUSIPCO, 2009.

[8] Y. K. Yılmaz, A. T. Cemgil, Algorithms for probabilistic latent tensor factorization, Signal Processing(2011),doi:10.1016/j.sigpro.2011.09.033.

[9] C. E. McCulloch, S. R. Searle, Generalized, Linear, and Mixed Models, Wiley, 2001.

[10] C. E. McCulloch, J. A. Nelder, Generalized Linear Models, 2nd Edition, Chapman and Hall, 1989.

[11] R. Kaas, Compound poisson distributions and glm's, tweedie's distribution, Tech. rep., Lecture, Royal Flemish Academy of Belgium for Science and the Arts, (2005).

[12] A. Cichocki, R. Zdunek, A. H. Phan, S. Amari, Nonnegative Matrix and Tensor Factorization, Wiley, 2009.

[13] J. R. Magnus, H. Neudecker, Matrix Differential Calculus with Applications in Statistics and Econometrics, 3rd Edition, Wiley, 2007.

[14] M. Wainwright, M. I. Jordan, Graphical models, exponential families, and variational inference, Foundations and Trends in Machine Learning 1 (2008) 1–305.

[15] D. D. Lee, H. S. Seung, Algorithms for non-negative matrix factorization, in: NIPS, Vol. 13, 2001, pp. 556–562.

[16] M. Marcus, H. Minc, A Survey of Matrix Theory and Matrix Inequalities, Dover, 1992.

[17] R. Salakhutdinov, A. Mnih, Probabilistic matrix factorization, in: Advances in Neural Information Processing Systems, Vol. 20, 2008.

[18] G. H. Golub, C. F. V. Loan, Matrix computations, 3rd Edition, Johns Hopkins UP, 1996.

[19] P. Smaragdis, J. C. Brown, Non-negative matrix factorization for polyphonic music transcription, in: WASPAA, 2003, pp. 177–180.

